# Gaussian Fields for Approximate Inference in Layered Sigmoid Belief Networks

**David Barber***
Stichting Neurale Netwerken
Medical Physics and Biophysics
Nijmegen University, The Netherlands
barberd@aston.ac.uk

**Peter Sollich**
Department of Mathematics
King's College, University of London
London WC2R 2LS, U.K.
peter.sollich@kcl.ac.uk

## Abstract

Layered Sigmoid Belief Networks are directed graphical models in which the local conditional probabilities are parameterised by weighted sums of parental states. Learning and inference in such networks are generally intractable, and approximations need to be considered. Progress in *learning* these networks has been made by using variational procedures. We demonstrate, however, that variational procedures can be inappropriate for the equally important issue of *inference* - that is, calculating marginals of the network. We introduce an alternative procedure, based on assuming that the weighted input to a node is approximately Gaussian distributed. Our approach goes beyond previous Gaussian field assumptions in that we take into account correlations between parents of nodes. This procedure is specialized for calculating marginals and is significantly faster and simpler than the variational procedure.

## 1  Introduction

Layered Sigmoid Belief Networks [1] are directed graphical models [2] in which the local conditional probabilities are parameterised by weighted sums of parental states, see fig(1). This is a graphical representation of a distribution over a set of binary variables $s_i \in \{0, 1\}$. Typically, one supposes that the states of the nodes at the bottom of the network are *generated* by states in previous layers. Whilst, in principle, there is no restriction on the number of nodes in any layer, typically, one considers structures similar to the "fan out" in fig(1) in which higher level layers provide an "explanation" for patterns generated in lower layers. Such graphical models are attractive since they correspond to layers of information processors, of potentially increasing complexity. Unfortunately, learning and inference in such networks is generally intractable, and approximations need to be considered. Progress in learning has been made by using variational procedures [3, 4, 5]. However, another crucial aspect remains inference [2]. That is, given some evidence (or none), calculate the marginal of a variable, conditional on this evidence. This assumes that we have found a suitable network from some learning procedure, and now wish

to query this network. Whilst the variational procedure is attractive for learning, since it generally provides a bound on the likelihood of the visible units, we demonstrate that it may not always be equally appropriate for the inference problem.

A directed graphical model defines a distribution over a set of variables $\mathbf{s} = (s_1 \ldots s_n)$ that factorises into the local conditional distributions,

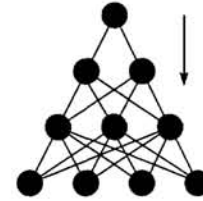

$$p\left(s_1 \ldots s_n\right) = \prod_{i=1}^{n} p\left(s_i | \pi_i\right) \qquad (1)$$

where $\pi_i$ denotes the parent nodes of node $i$. In a layered network, these are the nodes in the proceeding layer that feed into node $i$. In a sigmoid belief network the local probabilities are defined as

Figure 1: A Layered Sigmoid Belief Network

$$p\left(s_i = 1 | \pi_i\right) = \sigma\left(\sum_j w_{ij} s_j + \theta_i\right) = \sigma\left(h_i\right) \qquad (2)$$

where the "field" at node $i$ is defined as $h_i = \sum_j w_{ij} s_j + \theta_i$ and $\sigma(h) = 1/(1 + e^{-h})$. $w_{ij}$ is the strength of the connection between node $i$ and its parent node $j$; if $j$ is not a parent of $i$ we set $w_{ij} = 0$. $\theta_i$ is a bias term that gives a parent-independent bias to the state of node $i$.

We are interested in inference - in particular, calculating marginals of the network for cases with and without evidential nodes. In section (2) we describe how to approximate the quantities $p(s_i = 1)$ and discuss in section (2.1) why our method can improve on the standard variational mean field theory. Conditional marginals, such as $p(s_i = 1 | s_j = 1, s_k = 0)$ are considered in section (3).

## 2   Gaussian Field Distributions

Under the 0/1 coding for the variables $s_i$, the mean of a variable, $m_i$ is given by the probability that it is in state 1. Using the fact from (2) that the local conditional distribution of node $i$ is dependent on its parents *only* through its field $h_i$, we have

$$m_i = p\left(s_i = 1\right) = \int p\left(s_i = 1 | h_i\right) p\left(h_i\right) dh_i \equiv \left\langle \sigma\left(h_i\right) \right\rangle_{p(h_i)} \qquad (3)$$

where we use the notation $\left\langle (\cdot) \right\rangle_p$ to denote an average with respect to the distribution $p$. If there are many parents of node $i$, a reasonable assumption is that the distribution of the field $h_i$ will be Gaussian, $p(h_i) \approx N\left(\mu_i, \sigma_i^2\right)$. Under this Gaussian Field (GF) assumption, we need to work out the mean and variance, which are given by

$$\mu_i = \left\langle h_i \right\rangle = \sum_j w_{ij} \left\langle s_j \right\rangle + \theta_i = \sum_j w_{ij} m_j + \theta_i \qquad (4)$$

$$\sigma_i^2 = \left\langle \left(\Delta h_i\right)^2 \right\rangle = \sum_{j,k} w_{ij} w_{ik} R_{jk} \qquad (5)$$

where $R_{jk} = \left\langle \Delta s_j \Delta s_k \right\rangle$. We use the notation $\Delta\left(\cdot\right) \equiv \left(\cdot\right) - \left\langle \left(\cdot\right) \right\rangle$.

The diagonal terms of the node covariance matrix are $R_{ii} = m_i(1 - m_i)$. In contrast to previous studies, we include off diagonal terms in the calculation of $R$ [4]. From

(5) we only need to find correlations between parents $i$ and $j$ of a node. These are easy to calculate in the layered networks that we are considering, because neither $i$ nor $j$ is a descendant of the other:

$$R_{ij} = p(s_i = 1, s_j = 1) - m_i m_j \tag{6}$$

$$= \int p(s_i = 1|h_i) p(s_j = 1|h_j) p(h_i, h_j) dh - m_i m_j \tag{7}$$

$$= \langle \sigma(h_i) \sigma(h_j) \rangle_{p(h_i, h_j)} - m_i m_j \tag{8}$$

Assuming that the joint distribution $p(h_i, h_j)$ is Gaussian, we again need its mean and covariance, given by

$$\boldsymbol{\mu}^T = (\langle h_i \rangle, \langle h_j \rangle) = \left( \sum_k w_{ik} m_k + \theta_i, \sum_l w_{jl} m_l + \theta_j \right) \tag{9}$$

$$\Sigma_{ij} = \langle \Delta h_i \Delta h_j \rangle = \sum_{kl} w_{ik} w_{jl} \langle \Delta s_k \Delta s_l \rangle = \sum_{kl} w_{ik} w_{jl} R_{kl} \tag{10}$$

Under this scheme, we have a closed set of equations, (4,5,8,10) for the means $m_i$ and covariance matrix $R_{ij}$ which can be solved by forward propagation of the equations. That is, we start from nodes without parents, and then consider the next layer of nodes, repeating the procedure until a full sweep through the network has been completed. The one and two dimensional field averages, equations (3) and (8), are computed using Gaussian Quadrature. This results in an extremely fast procedure for approximating the marginals $m_i$, requiring only a single sweep through the network.

Our approach is related to that of [6] by the common motivating assumption that each node has a large number of parents. This is used in [6] to obtain actual bounds on quantities of interest such as joint marginals. Our approach does not give bounds. Its advantage, however, is that it allows fluctuations in the fields $h_i$, which are effectively excluded in [6] by the assumed scaling of the weights $w_{ij}$ with the number of parents per node.

## 2.1 Relation to Variational Mean Field Theory

In the variational approach, one fits a tractable approximating distribution $Q$ to the SBN. Taking $Q$ factorised, $Q(\mathbf{s}) = \prod_i m_i^{s_i} (1 - m_i)^{1-s_i}$ we have the bound

$$\ln p(s_1 \ldots s_n) \geq \sum_i \{ -m_i \ln m_i - (1 - m_i) \ln (1 - m_i) \}$$

$$+ \sum_i \left\{ \sum_j m_i w_{ij} m_j + \theta_i m_i - \langle \ln (1 + e^{h_i}) \rangle_Q \right\} \tag{11}$$

The final term in (11) causes some difficulty even in the case in which $Q$ is a factorised model. Formally, this is because this term does not have the same graphical structure as the tractable model $Q$. One way around around this difficulty is to employ a further bound, with associated variational parameters [7]. Another approach is to make the Gaussian assumption for the field $h_i$ as in section (2). Because $Q$ is factorised, corresponding to a diagonal correlation matrix $R$, this gives [4]

$$\langle \ln (1 + e^{h_i}) \rangle_Q \approx \langle \ln (1 + e^{h_i}) \rangle_{N(\mu_i, \sigma_i^2)} \tag{12}$$

where $\mu_i = \sum_j w_{ij} m_j + \theta_i$ and $\sigma_i^2 = \sum_j w_{ij}^2 m_j (1 - m_j)$. Note that this is a one dimensional integral of a smooth function. In contrast to [4] we therefore evaluate this quantity using Gaussian Quadrature. This has the advantage that no extra variational parameters need to be introduced. Technically, the assumption of a Gaussian field distribution means that (11) is no longer a bound. Nevertheless, in practice it is found that this has little effect on the quality of the resulting solution. In our implementation of the variational approach, we find the optimal parameters $m_i$ by maximising the above equation for each component $m_i$ separately, cycling through the nodes until the parameters $m_i$ do not change by more than $10^{-10}$. This is repeated 5 times, and the solution with the highest bound score is chosen. Note that these equations cannot be solved by forward propagation alone since the final term contains contributions from all the nodes in the network. This is in contrast to the GF approach of section (2). Finding appropriate parameters $m_i$ by the variational approach is therefore rather slower than using the GF method.

In arriving at the above equations, we have made two assumptions. The first is that the intractable distribution is well approximated by a factorised model. The second is that the field distribution is Gaussian. The first step is necessary in order to obtain a bound on the likelihood of the model (although this is slightly compromised by the Gaussian field assumption). In the GF approach we dispense with this assumption of an effectively factorised network (partially because if we are only interested in inference, a bound on the model likelihood is less relevant). The GF method may therefore prove useful for a broader class of networks than the variational approach.

## 2.2   Results for unconditional marginals

We compared three procedures for estimating the conditional values $p(s_i = 1)$ for all the nodes in the network, namely the variational theory, as described in section (2.1), the diagonal Gaussian field theory, and the non-diagonal Gaussian field theory which includes correlation effects between parents. Results for small weight values $w_{ij}$ are shown in fig(2). In this case, all three methods perform reasonably well, although there is a significant improvement in using the GF methods over the variational procedure; parental correlations are not important (compare figs(2b) and (2c)). In fig(3) the weights and biases are chosen such that the exact mean variables $m_i$ are roughly 0.5 with non-trivial correlation effects between parents. Note that the variational mean field theory now provides a poor solution, whereas the GF methods are relatively accurate. The effect of using the non-diagonal $R$ terms is beneficial, although not dramatically so.

## 3   Calculating Conditional Marginals

We consider now how to calculate conditional marginals, given some evidential nodes. (In contrast to [6], any set of nodes in the network, not just output nodes, can be considered evidential.) We write the evidence in the following manner

$$E = \{s_{c_1} = S_{c_1}, \ldots s_{c_n} = S_{c_n}\} = \{E_{c_1} \ldots E_{c_n}\}$$

The quantities that we are interested in are conditional marginals which, from Bayes rule are related to the joint distribution by

$$p(s_i = 1|E) = \frac{p(s_i = 1, E)}{p(s_i = 0, E) + p(s_i = 1, E)} \tag{13}$$

That is, provided that we have a procedure for estimating joint marginals, we can obtain conditional marginals too. Without loss of generality, we therefore consider

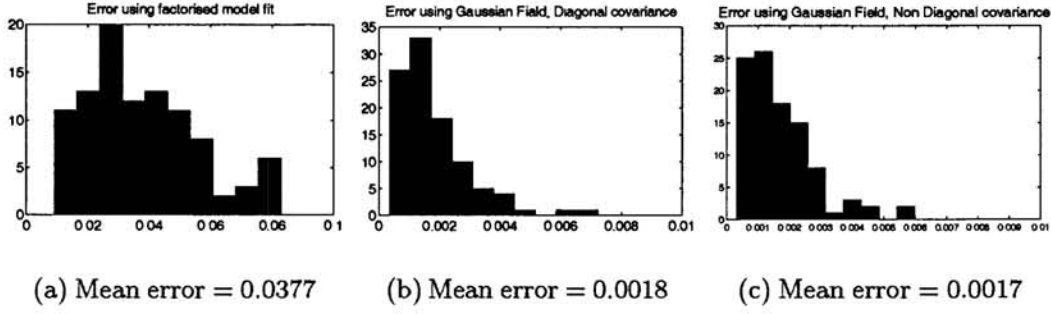

(a) Mean error = 0.0377     (b) Mean error = 0.0018     (c) Mean error = 0.0017

Figure 2: Error in approximating $p(s_i = 1)$ for the network in fig(1), averaged over all the nodes in the network. In each of 100 trials, weights were drawn from a zero mean, unit variance Gaussian; biases were set to 0. Note the different scale in (b) and (c). In (a) we use the variational procedure with a factorised $Q$, as in section (2.1). In (b) we use the Gaussian field equations, assuming a diagonal covariance matrix $R$. This procedure was repeated in (c) including correlations between parents.

$E^+ = E \cup \{s_i = 1\}$, which then contains $n + 1$ "evidential" variables. That is, the desired marginal variable is absorbed into the evidence set. For convenience, we then split the nodes into two sets, those containing the evidential or "clamped" nodes, $C$, and the remaining "free" nodes $F$. The joint evidence is then given by

$$p(E^+) = \sum_{s_F} p\left(E_{c_1}, \ldots E_{c_{n+1}}, s_{f_1}, \ldots s_{f_m}\right) \tag{14}$$

$$= \sum_{s_F} p\left(E_{c_1}|\pi_{c_1}^*\right) \ldots p\left(E_{n+1}|\pi_{c_{n+1}}^*\right) p\left(s_{f_1}|\pi_{f_1}^*\right) \ldots p\left(s_{f_m}|\pi_{f_m}^*\right) \tag{15}$$

where $\pi_i^*$ are the parents of node $i$, with any evidential parental nodes set to their values as specified in $E^+$. In the sigmoid belief network

$$p(E_k|\pi_k^*) = \sigma\left((2S_k - 1)\left(\sum_i w_{ki}s_i^* + \theta_k\right)\right), \quad s_i^* = \begin{cases} S_i, & \text{if } i \text{ is an evidential node} \\ s_i, & \text{otherwise} \end{cases} \tag{16}$$

$p(E_k|\pi_k^*)$ is therefore determined by the distribution of the field $h_k^* = \sum_i w_{ki}s_i^* + \theta_k$. Examining (15), we see that the product over the "free" nodes defines a SBN in which the local probability distributions are given by those of the original network, but with any evidential parental nodes clamped to their evidence values. Therefore,

$$p\left(E^+\right) = \left\langle \prod_{i=1}^{n+1} \sigma\left((2S_{c_i} - 1)h_{c_i}^*\right) \right\rangle_{p\left(h_{c_1}^* \ldots h_{c_{n+1}}^*\right)} \tag{17}$$

Consistent with our previous assumptions, we assume that the distribution of the fields $\mathbf{h}^* = \left(h_{c_1}^* \ldots h_{c_{n+1}}^*\right)$ is jointly Gaussian. We can then find the mean and covariance matrix for the distribution of $\mathbf{h}^*$ by repeating the calculation of section (2) in which evidential nodes have been clamped to their evidence values. Once this Gaussian has been determined, it can be used in (17) to determine $p(E^+)$. Gaussian averages of products of sigmoids are calculated by drawing 1000 samples from the Gaussian over which we wish to integrate[1]. Note that if there are evidential nodes

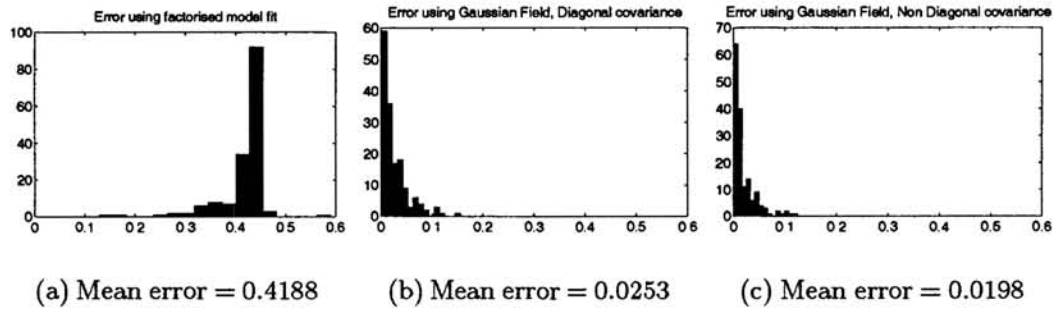

(a) Mean error = 0.4188     (b) Mean error = 0.0253     (c) Mean error = 0.0198

Figure 3: All weights are set to uniformly from 0 to 50. Biases are set to -0.5 of the summed parental weights plus a uniform random number from -2.5 to 2.5. The root node is set to be 1 with probability 0.5. This has the effect of making all the nodes in the exact network roughly 0.5 in mean, with non-negligible correlations between parental nodes. 160 simulations were made.

in different layers, we require the correlations between their fields $h$ to evaluate (17). Such 'inter-layer' correlations were not required in section (2), and to be able to use the same calculational scheme we simply neglect them. (We leave a study of the effects of this assumption for future work.) The average in (17) then factors into groups, where each group contains evidential terms in a particular layer.

The conditional marginal for node $i$ is obtained from repeating the above procedure in which the desired marginal node is clamped to its opposite value, and then using these results in (13). The above procedure is repeated for each conditional marginal that we are interested in. Although this may seem computationally expensive, the marginal for each node is computed quickly, since the equations are solved by one forward propagation sweep only.

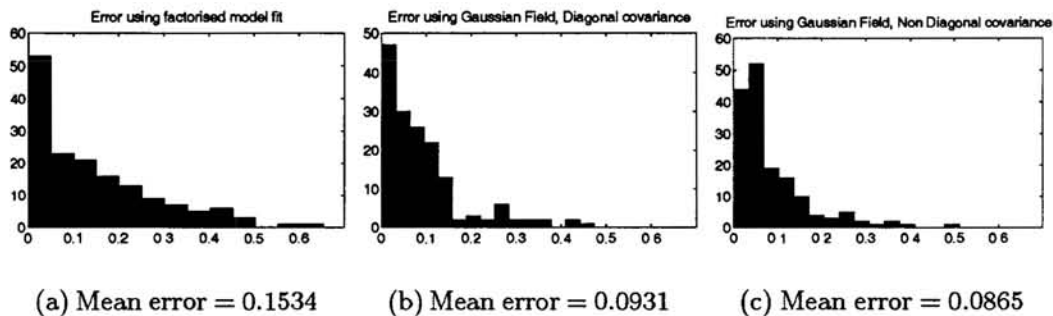

(a) Mean error = 0.1534     (b) Mean error = 0.0931     (c) Mean error = 0.0865

Figure 4: Estimating the conditional marginal of the top node being in state 1, given that the four bottom nodes are in state 1. Weights were drawn from a zero mean Gaussian with variance 5, with biases set to -0.5 the summed parental weights plus a uniform random number from -2.5 to 2.5. Results of 160 simulations.

## 3.1 Results for conditional marginals

We used the same structure as in the previous experiments, as shown in fig(1). We are interested here in calculating the probability that the top node is in state 1,

given that the four bottom nodes are in state 1. Weights were chosen from a zero mean Gaussian with variance 5. Biases were set to negative half of the summed parent weights, plus a uniform random value from -2.5 to 2.5. Correlation effects in these networks are not as strong as in the experiments in section (2.2), although the improvement of the GF theory over the variational theory seen in fig(4) remains clear. The improvement from the off diagonal terms in $R$ is minimal.

# 4  Conclusion

Despite their appropriateness for learning, variational methods may not be equally suited to inference, making more tailored methods attractive. We have considered an approximation procedure that is based on assuming that the distribution of the weighted input to a node is approximately Gaussian. Correlation effects between parents of a node were taken into account to improve the Gaussian theory, although in our examples this gave only relatively modest improvements.

The variational mean field theory performs poorly in networks with strong correlation effects between nodes. On the other hand, one may conjecture that the Gaussian Field approach will not generally perform catastrophically worse than the factorised variational mean field theory. One advantage of the variational theory is the presence of an objective function against which competing solutions can be compared. However, finding an optimum solution for the mean parameters $m_i$ from this function is numerically complex. Since the Gaussian Field theory is extremely fast to solve, an interesting compromise might be to prime the variational solution with the results from the Gaussian Field theory.

**Acknowledgments**

DB would like to thank Bert Kappen and Wim Wiegerinck for stimulating and helpful discussions. PS thanks the Royal Society for financial support.

[1] R. Neal. Connectionist learning of Belief Networks. *Artificial Intelligence*, 56:71–113, 1992.

[2] E. Castillo, J. M. Gutierrez, and A. S. Hadi. *Expert Systems and Probabilistic Network Models*. Springer, 1997.

[3] M. I. Jordan, Z. Gharamani, T. S. Jaakola, and L. K. Saul. An Introduction to Variational Methods for Graphical Models. In M. I. Jordan, editor, *Learning in Graphical Models*, pages 105–161. Kluwer, 1998.

[4] L. Saul and M. I. Jordan. A mean field learning algorithm for unsupervised neural networks. In M. I. Jordan, editor, *Learning in Graphical Models*, 1998.

[5] D. Barber and W Wiegerinck. Tractable variational structures for approximating graphical models. In M.S. Kearns, S.A. Solla, and D.A. Cohn, editors, *Advances in Neural Information Processing Systems NIPS 11*. MIT Press, 1999.

[6] M. Kearns and L. Saul. Inference in Multilayer Networks via Large Deviation Bounds. In *Advances in Neural Information Processing Systems NIPS 11*, 1999.

[7] L. K. Saul, T. Jaakkola, and M. I. Jordan. Mean Field Theory for Sigmoid Belief Networks. *Journal of Artificial Intelligence Research*, 4:61–76, 1996.